# Maximum Margin Multi-Instance Learning

**Hua Wang**
Computer Science and Engineering
University of Texas at Arlington
huawangcs@gmail.com

**Heng Huang**
Computer Science and Engineering
University of Texas at Arlington
heng@uta.edu

**Farhad Kamangar**
Computer Science and Engineering
University of Texas at Arlington
kamangar@uta.edu

**Feiping Nie**
Computer Science and Engineering
University of Texas at Arlington
feipingnie@gmail.com

**Chris Ding**
Computer Science and Engineering
University of Texas at Arlington
chqding@uta.edu

## Abstract

Multi-instance learning (MIL) considers input as bags of instances, in which labels are assigned to the bags. MIL is useful in many real-world applications. For example, in image categorization semantic meanings (labels) of an image mostly arise from its regions (instances) instead of the entire image (bag). Existing MIL methods typically build their models using the *Bag-to-Bag (B2B)* distance, which are often computationally expensive and may not truly reflect the semantic similarities. To tackle this, in this paper we approach MIL problems from a new perspective using the *Class-to-Bag (C2B)* distance, which directly assesses the relationships between the classes and the bags. Taking into account the two major challenges in MIL, high heterogeneity on data and weak label association, we propose a novel Maximum Margin Multi-Instance Learning (M$^3$I) approach to parameterize the C2B distance by introducing the class specific distance metrics and the locally adaptive significance coefficients. We apply our new approach to the automatic image categorization tasks on three (one single-label and two multi-label) benchmark data sets. Extensive experiments have demonstrated promising results that validate the proposed method.

## 1  Introduction

Traditional image categorization methods usually consider an image as one indiscrete entity, which, however, neglects an important fact that the semantic meanings (labels) of an image mostly arise from its constituent regions, but not the entire image. For example, the labels "person" and "car" associated with the query image in Figure 1 are only characterized by the regions in two bounding boxes, respectively, rather than the whole image. Therefore, modeling the relationships between labels and regions (instead of the entire image) could potentially reduce the noise in the corresponding feature space, and the learned semantic models could be more accurate.

In recent years, image representation techniques using semi-local, or patch-based, features, such as SIFT, have demonstrated some of the best performance in image retrieval and object recognition applications. These algorithms choose a set of patches in an image, and for each patch compute

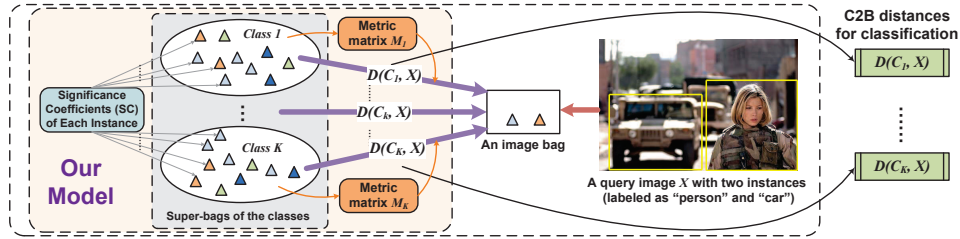

Figure 1: Diagram of the proposed $M^3I$ approach. Our task is to learn class specific distance metrics $M_k$ and significance coefficients $w_k^j$ from the training data, with which we compute the C2B distances from the classes to a query image $X$ for classification.

a fixed-length feature vector. This gives a set of vectors per image, where the size of the set can vary from image to image. Armed with these patch-based features, image categorization is recently formulated as a *multi-instance learning* (MIL) task [1–6]. Under the framework of MIL, an image is viewed as a *bag*, which contains a number of *instances* corresponding to the regions in the image. If any of these instances is related to a semantic concept, the image will be associated with the corresponding label. The goal of MIL is to construct a learner to classify unseen image bags.

## 1.1 Learning Class-to-Bag (C2B) distance for multi-instance data

In MIL data objects are represented as bags of instances, therefore the distance between the objects is a set-to-set distance. Compared to traditional single-instance data that use vector distance such as Euclidean distance, estimating the *Bag-to-Bag (B2B)* distance in MIL is more challenging [7, 8]. In addition, the B2B distances often do not truly reflect the semantic similarities [9]. For example, two images containing one common object may also have other visually incompatible objects, which makes these two images less similar in terms of the B2B distance. Therefore, instead of measuring the similarities between bags, in this paper we approach MIL from a new perspective using the *Class-to-Bag (C2B)* distance, which assesses the relationships between the classes and the bags.

Measuring the distance between images (bags) and classes was first introduced in [9] for object recognition, which used the *Bag-to-Class (B2C) distance* instead of the C2B distance. Given a triplet constraint $(i, p, n)$ that image $i$ is more relevant to class $p$ than it is to class $n$, the C2B distance formulates this as $D_{pi} < D_{ni}$, while the B2C distance formulates this as $D_{ip} < D_{in}$. It seems these two formulations are similar, however, they are different when learning parameterized distance, the main goal of this paper. To be more specific, for the C2B distance we only need to parameterize training instances, which are available during the training phase. In contrast, for the B2C distance, parameterizing instances in query images has to be involved, which is not always feasible because we typically do not know them beforehand. This difference will become more clear shortly when we mathematically define the C2B distance.

## 1.2 Challenges and opportunities of MIL

Multi-instance data are different from traditional single-instance data, which bring new opportunities to improve the classification performance, though together with more challenges. We first explore these challenges, as well as to find opportunities to enhance the C2B distance introduced above.

**Learning class specific distance metrics.** Due to the well-known semantic gap between low-level visual features and high-level semantic concepts [10], choosing an appropriate distance metric plays an important role in establishing an effective image categorization system, as well as other general MIL models. Existing metric learning methods [5,6] for multi-instance data only learned one global metric for an entire data set. However, multi-instance data by nature are highly heterogeneous, thus a homogeneous distance metric may not suffice to characterize different classes of objects in a same data set. For example, in Figure 1 the shape and color characterizations of a person are definitely different from those of a car. To this end, we consider to learn multiple distance metrics, one for each class, for a multi-instance data set to capture the correlations among the features within each object category. The metrics are learned simultaneously by forming a maximum margin optimization problem with the constraints that the C2B distances from the correct classes of an training object to it should be less than the distances from other classes to it by a margin.

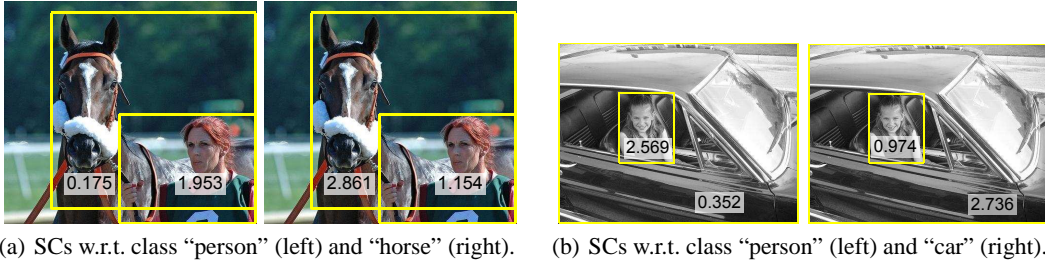

(a) SCs w.r.t. class "person" (left) and "horse" (right).   (b) SCs w.r.t. class "person" (left) and "car" (right).

Figure 2: The learned SCs of the instances in a same image when they serve as training samples for different classes. For example, in Figure 2(a) the SC of the horse instance in class "person" is 0.175, whereas its SC in class "horse" is 2.861. As a result, the horse instance contributes a lot in the C2B distance from class "horse" to a query image, while having much less impact in the C2B distance from class "person" to a query image.

**Learning locally adaptive C2B distance.** Different from the classification problems for traditional single-instance data, in MIL the classes are weakly associated to the bags, *i.e.*, a label is assigned to a bag as long as one of its instance belongs to the class. As a result, although a bag is associated with a class, some, or even most, of its instances may not be truly related to the class. For example, in the query image in Figure 1, the instance in the left bounding box does not contribute to the label "person". Intuitively, the instances in a super-bag of a class should not contribute equally in predicting labels for a query image. Instead, they should be properly weighted. With this recognition, we formulate another maximum margin optimization problem to learn multiple weights for a training instance, one for each of its labeled classes. The resulted weight reflects the relative importance of the training instance with respect to a class, which we call as *Significance Coefficient (SC)* . Ideally, the SC of an instance with respect to its true belonging class should be large, whereas its SC with respect other classes should be small. In Figure 2, we show the learned SCs for the instances in some images when they serve as training samples for their labeled classes. Because the image in Figure 2(a) has two labels, this image, thereby its two instances, serves as a training sample for both class "person" (left) and "horse" (right). Although the learned SC of the horse instance is very low when it is in the super-bag of "person" (0.175) as in the left panel of Figure 2(a), its SC (2.861) is relatively high when it is in the super-bag of "horse", its true belonging class, as in right panel of Figure 2(a). The same observations can also be seen in the rest examples, which are perfectly in accordance with our expectations.

With the above two enhancements to C2B distance, the class specific metrics and SCs, the two difficulties in MIL are addressed. Because these two components of the proposed approach are learned from two maximum margin optimization problems, we call the proposed approach as Maximum Margin Multi-Instance Learning ($M^3I$) approach, which is schematically illustrated in Figure 1.

## 2   Learning C2B distance for multi-instance data via $M^3I$ approach

In this section, we first briefly formalize the MIL problem and the C2B distance for a multi-instance data set, where we provide the notations used in this paper. Then we gradually develop the proposed $M^3I$ approach to incorporate the class specific distance metrics and the locally adaptive SCs into the C2B distance, together with its learning algorithms.

**Problem formalization of MIL.** Given a multi-instance data set with $K$ classes and $N$ training bags, we denote the training set by $\mathcal{D} = \{(X_i, \mathbf{y}_i)\}_{i=1}^{N}$. Each $X_i = \{\mathbf{x}_i^1, \ldots, \mathbf{x}_i^{n_i}\}$ is a bag of $n_i$ instances, where $\mathbf{x}_i^j \in \mathbb{R}^p$ is a vector of $p$ dimensions. The class assignment indicator $\mathbf{y}_i \in \{0, 1\}^K$ is a binary vector, with $\mathbf{y}_i(k) = 1$ indicating that bag $X_i$ belongs to the $k$-th class and $\mathbf{y}_i(k) = 0$ otherwise. We write $Y = [\mathbf{y}_1, \ldots, \mathbf{y}_N]^T$. If $\sum_{k=1}^{K} Y_{ik} = 1$, *i.e.*, each bag belongs to exactly one class, the data set is a single-label data set; if $\sum_{k=1}^{K} Y_{ik} \geq 1$, *i.e.*, each bag may be associated with more than one class label, the data set is a multi-label data set [11–14]. In the setting of MIL, we assume that (I) bag $X$ is assigned to the $k$-th class $\Longleftrightarrow$ at least one instance of $X$ belongs to the $k$-th class; and (II) bag $X$ is not assigned to the $k$-th class $\Longleftrightarrow$ no instance of $X$ belongs to the $k$-th class. Our task is to learn from $\mathcal{D}$ a classifier that is able to predict labels for a new query bag.

For convenience, we denote $\mathcal{P}(X_i)$ as the classes that bag $X_i$ belongs to (positive classes), and $\mathcal{N}(X_i)$ as the classes that $X_i$ does not belong to (negative classes).

**C2B distance in MIL.** In order to compute the C2B distance, we represent every class as a super-bag, *i.e.*, a set consisting of all the instances in every bag belonging to a class:

$$S_k = \left\{ \mathbf{s}_k^1, \ldots, \mathbf{s}_k^{m_k} \right\} = \left\{ \mathbf{x}_i^j \mid Y_{ik} = 1 \right\}, \tag{1}$$

where $\mathbf{s}_k^j$ is an instance of $S_k$ that comes from one of the training bags belonging to the $k$-th class, and $m_k = \sum_{\{i|Y_{ik}=1\}} n_i$ is the total number of the instances in $S_k$. Note that, in single-label data where each bag belongs to only one class, we have $S_k \cap S_l = \varnothing \;\; (\forall\, k \neq l)$ and $\sum_{k=1}^K m_k = \sum_{i=1}^N n_i$. In multi-label data where each bag (thereby each instance) may belong to more than one class [11–14], we have $S_k \cap S_l \neq \varnothing \;\; (\forall\, k \neq l)$ and $\sum_{k=1}^K m_k \geq \sum_{i=1}^N n_i$, *i.e.*, different super-bags may overlap and one $\mathbf{x}_i^j$ may appear in multiple super-bags.

The elementary distance from an instance in a super-bag to a bag is defined as the distance between this instance and its nearest neighbor instance in the bag:

$$d\left( \mathbf{s}_k^j, X_i \right) = \left\| \mathbf{s}_k^j - \tilde{\mathbf{s}}_k^j \right\|_M^2, \tag{2}$$

where $\tilde{\mathbf{s}}_k^j$ is the nearest neighbor instance of $\mathbf{s}_k^j$ in $X_i$.

Then we compute the C2B distance from a super-bag $S_k$ to a data bag $X_i$ as following:

$$D\left( S_k, X_i \right) = \sum_{j=1}^{m_k} d\left( \mathbf{s}_k^j, X_i \right) = \sum_{j=1}^{m_k} \left\| \mathbf{s}_k^j - \tilde{\mathbf{s}}_k^j \right\|^2. \tag{3}$$

## 2.1 Parameterized C2B distance of the M$^3$I approach

Because the C2B distance defined in Eq. (3) does not take into account the challenging properties of multi-instance data as discussed in Section 1.2, we further develop it in the rest of this subsection.

**Class specific distance metrics.** The C2B distance defined in Eq. (3) is a Euclidean distance, which is independent of the input data. In order to capture the second-order statistics of the input data that could potentially improve the subsequent classification [5, 6], we consider to use the Mahalanobis distance with an appropriate distance metric. With the recognition of the high heterogeneity in multi-instance data, instead of learning a global distance metric as in existing works [5, 6], we propose to learn $K$ different class specific distance metrics $\{M_k\}_{k=1}^K \subset \mathbb{R}^{p \times p}$, one for each class. Note that, using class specific distance metrics is only feasible with the distance between classes and bags (either C2B or B2C distance), because we are only concerned with intra-class distances. In contrast, traditional B2B distance needs to compute the distances between bags belonging to different classes involving inter-class distance metrics, which inevitably complicates the problem.

Specifically, instead of using Eq. (3), we compute C2B distance using the Mahalanobis distance as:

$$D\left( S_k, X_i \right) = \sum_{j=1}^{m_k} \left[ \left( \mathbf{s}_k^j - \tilde{\mathbf{s}}_k^j \right)^T M_k \left( \mathbf{s}_k^j - \tilde{\mathbf{s}}_k^j \right) \right]. \tag{4}$$

**Locally adaptive C2B distance.** Now we further develop the C2B distance defined in Eq. (4) to address the labeling ambiguity in multi-instance scenarios. We propose a locally adaptive C2B distance by weighting the instances in a super-bag upon their relevance to the corresponding class.

Due to the weak association between the instances and the bag labels, not every instance in a super-bag of a class truly characterizes the corresponding semantic concept. For example, in Figure 2(a) the region for the horse object is in the super-bag of "person" class, because the entire image is labeled with both "person" and "horse". As a result, intuitively, we should give a smaller, or even no, weight to the horse instance when determining whether to assign "person" label to a query

image; and give it a higher weight when deciding "horse" label. To be more precise, let $w_k^j$ be the weight associated with $\mathbf{s}_k^j$, we wish to learn the C2B distance as following:

$$D\left(S_k, X_i\right) = \sum_{j=1}^{m_k} \left[ w_k^j \left(\mathbf{s}_k^j - \tilde{\mathbf{s}}_k^j\right)^T M_k \left(\mathbf{s}_k^j - \tilde{\mathbf{s}}_k^j\right) \right] \quad . \tag{5}$$

Because $w_k^j$ reflects the relative importance of instance $\mathbf{s}_k^j$ when determining the label for the $k$-th class, we call it as the "Significance Coefficient (SC)" of $\mathbf{s}_k^j$.

## 2.2 Procedures to learn $M_k$ and $w_k^j$

Given the parameterized C2B distance defined in Eq. (5) for a multi-instance data set, our learning objects are the two sets of variables $M_k$ and $w_k^j$. Motivated by metric learning from relative comparisons [15–17], we learn $M_k$ and $w_k^j$ by constraining that the C2B distances from the true belonging classes of bag $X_i$ to it are smaller than the distances from any other classes to it by a margin:

$$\forall\, p \in \mathcal{P}\left(X_i\right),\ n \in \mathcal{N}\left(X_i\right): \quad D\left(S_n, X_i\right) - D\left(S_p, X_i\right) \geq 1 - \xi_{ipn}, \tag{6}$$

where $\xi_{ipn}$ is a slack variable because the constraints usually can not be completely satisfied in real world data. Therefore, $\xi_{ipn}$ measures the deviation from the strict constraint for the triplet $(i, p, n)$. In the following, we formulate two maximum margin optimization problems to learn the two sets of target variables $M_k$ and $w_k^j$, one for each of them.

**Optimizing $M_k$.** First we fix $w_k^j$ to optimize $M_k$. To avoid over-fitting, as in support vector machine (SVM), we minimize the overall C2B distances from $X_i$'s associated classes to itself and the total amount of slack. Specifically, we solve the following convex optimization problem:

$$\min_{M_1,\ldots,M_K} \sum_{i,\ p\in\mathcal{P}(X_i),} D\left(S_p, X_i\right) + C \sum_{i,\ p\in\mathcal{P}(X_i),\ n\in\mathcal{N}(X_i)} \xi_{ipn},$$
$$s.t.\ \forall\, p \in \mathcal{P}\left(X_i\right), n \in \mathcal{N}\left(X_i\right): \xi_{ipn} \geq 0,\ D\left(S_n, X_i\right) - D\left(S_p, X_i\right) \geq 1 - \xi_{ipn}, \tag{7}$$
$$\forall\, k: M_k \succeq 0,$$

where $C$ is a trade-off parameter, acting same as in SVM. The optimization problem in Eq. (7) is a semi-definite programming (SDP) problem, which can be solved by standard SDP solvers. However, standard SDP solvers are computationally expensive. Therefore, we use the gradient descent SDP solver introduced in [18] to solve the problem.

**Optimizing $w_k^j$.** Then we fix $M_k$ to optimize $w_k^j$. Let $d_M\left(\mathbf{s}_k^j, X_i\right) = \left(\mathbf{s}_k^j - \tilde{\mathbf{s}}_k^j\right)^T M_k \left(\mathbf{s}_k^j - \tilde{\mathbf{s}}_k^j\right)$, we denote $\mathbf{d}_{ki} = \left[d_M\left(\mathbf{s}_k^1, X_i\right), \ldots, d_M\left(\mathbf{s}_k^{m_k}, X_i\right)\right]^T$. Let $\mathbf{w}_k = \left[w_k^1, \ldots, w_k^{m_k}\right]^T$, by the definition in Eq. (5) we rewrite Eq. (6) as following:

$$\mathbf{w}_n^T \mathbf{d}_{ni} - \mathbf{w}_p^T \mathbf{d}_{pi} \geq 1 - \xi_{ipn},\ \forall\, p \in \mathcal{P}\left(X_i\right), n \in \mathcal{N}\left(X_i\right). \tag{8}$$

In order to make use of the standard large-margin classification framework and simplify our derivation, following [17] we expand our notations. Let $\mathbf{w} = \left[\mathbf{w}_1^T, \ldots, \mathbf{w}_K^T\right]^T$, which is the concatenation of the class-specific weight vectors $\mathbf{w}_k$. Thus, each class-specific weight vector $\mathbf{w}_k$ corresponds a subrange of $\mathbf{w}$. Similarly, we expand the distance vectors and let $\mathbf{d}_{ipn}$ be a vector of the same length as $\mathbf{w}$, such that all its entries are 0 except the subranges corresponding to class $p$ and class $n$, which are set to be $-\mathbf{d}_{pi}$ and $\mathbf{d}_{ni}$ respectively. It is straightforward to to verify that $\mathbf{w}_n^T \mathbf{d}_{ni} - \mathbf{w}_p^T \mathbf{d}_{pi} = \mathbf{w}^T \mathbf{d}_{ipn}$. Thus Eq. (8) becomes:

$$\mathbf{w}^T \mathbf{d}_{ipn} \geq 1 - \xi_{ipn},\ \forall\, p \in \mathcal{P}\left(X_i\right), n \in \mathcal{N}\left(X_i\right) \quad . \tag{9}$$

Following the standard soft-margin SVM framework, we minimize the cumulative deviation over all triplet constraints $(i, p, n)$ and impose $\ell_2$-norm regularization on $\mathbf{w}$ as following:

$$\min_{\mathbf{w},\ \xi_{ipn}} \quad \frac{1}{2}\|\mathbf{w} - \mathbf{w}^{(0)}\|^2 + C \sum_{i, p\in\mathcal{P}(X_i), n\in\mathcal{N}(X_i)} \xi_{ipn}$$
$$s.t.\quad \forall\, i,\ p \in \mathcal{P}\left(X_i\right), n \in \mathcal{N}\left(X_i\right):\ \xi_{ipn} \geq 0,\ \mathbf{w}^T \mathbf{d}_{ipn} \geq 1 - \xi_{ipn}, \tag{10}$$
$$\forall\, j:\ \mathbf{w}\left(j\right) > 0,$$

where $C$ controls the tradeoff between the loss and regularization terms. The positivity constraint on the elements of $\mathbf{w}$ is due to the fact that our goal is to define a distance function which, by definition, is a positive definite operator. In addition, we also enforce a prior weight vector $\mathbf{w}^{(0)}$ in the objective. In standard SVM, all the entries of $\mathbf{w}$ are set as 0 as default. In our objective, however, we set all its entries to be 1, because we think all the instances are equally important if we have no prior training knowledge.

We solve Eq. (10) using the solver introduced in [17], which solves the dual problem by an accelerated iterative method. Upon solution, we obtain $\mathbf{w}$, which can be decomposed into the expected instance weights $w_k^j$ for every instance with respect to its labeled classes.

## 2.3 Label prediction using C2B distance

Solving the optimization problems in Eq. (7) and Eq. (10) for an input multi-instance data set $\mathcal{D}$, we obtain the learned class specific distance metrics $M_k$ $(1 \leq k \leq K)$ and the significance coefficients $w_k^j$ $(1 \leq k \leq K, 1 \leq j \leq m_k)$. Given a query bag $X$, upon the learned $M_k$ and $w_k^j$ we can compute the parameterized C2B distances $D(S_k, X)$ $(1 \leq k \leq K)$ from all the classes to the query bag using Eq. (5). Sorting $D(S_k, X)$, we can easily assign labels to the query bag.

For single-label multi-instance data, in which each bag belongs to one and only one class, we assign $X$ to the class with the minimum C2B distance, *i.e.*, $l(X) = \arg\min_k D(S_k, X)$.

For multi-label multi-instance data, in which one bag can be associated with more than one class label, we need a threshold to make prediction. For every class, we learn a threshold from the training data as $b_k = \sum_{i=1}^N Y_{ik} D(S_k, X_i) / \sum_{i=1}^K Y_{ik}$, which is the average of the C2B distances from the $k$-th class to all its training bags. Then we determine the class membership for $X$ using the following rule: assign $X$ to the $k$-th class if $D(S_k, X) < b_k$, and not otherwise.

# 3 Related works

**Learning B2C distance.** Due to the unsatisfactory performance and high computational complexity of machine vision models using B2B distance, a new perspective to compute B2C distance was presented in [9]. This non-parametric model does not involve training process. Though simple, it achieved promising results in object recognition. However, this method heavily relies on the large number of local features in the training and testing set. To address this, Wang *et al*. [18] further developed this method by introducing distance metrics, to achieve better results with a small amount of training. However, as discussed earlier in Section 1.1, B2C distance is hard to parameterize in may real world applications. To tackle this, we propose to use C2B distance for multi-instance data.

**Learning distance metric for MIL.** As demonstrated in literature [5,6], learning a distance metric from training data to maintain class information is beneficial for MIL. However, existing methods [5,6] learned only one global metric for a multi-instance data set, which is insufficient because the objects in multi-instance data by nature are highly heterogeneous. Recognizing this, we propose to learn multiple distance metrics, one for each class. [18] took a same perspective as us, though it does not clearly formalize image classification as a MIL task.

**Learning locally adaptive distance**. Due to the weak label association in MIL, instead of considering all the instance equally important, we assign locally adaptive SCs to every instance in training data. Locally adaptive distance was first introduced in [16,17] for B2B distance. Compared to it, the proposed C2B distance is more advantageous. First, C2B distance measures the relevance between a class and a bag, hence label prediction can be naturally made upon the resulted distance, whereas an additional classification step [16] or transformation [17] is required when B2B distance is used. Second, C2B distance directly assesses the relations between semantic concepts and image regions, hence it could narrow the gap between high-level semantic concepts and low-level visual features. Last, but not least, our C2B distance requires significantly less computation. Specifically, the triplet constraints used in C2B model are constructed between classes and bags whose number is $\mathcal{O}(NK^2)$, while those used in B2B model [16,17] are constructed between bags with number of $\mathcal{O}(N^3)$. As $N$ (bag number) is typically much larger than $K$ (class number), our approach is much more computationally efficient. Indeed, a constraint selection step was involved in [16,17].

Table 1: Performance comparison on Object Recognition data set.

| Methods | Accuracy |
|---|---|
| DD | $0.676 \pm 0.074$ |
| DD-SVM | $0.754 \pm 0.054$ |
| MIMLBoost | $0.793 \pm 0.033$ |
| MIMLSVM | $0.796 \pm 0.042$ |
| B2C | $0.672 \pm 0.013$ |
| B2C-M | $0.715 \pm 0.032$ |
| C2B | $0.797 \pm 0.015$ |
| C2B-M | $0.815 \pm 0.026$ |
| C2B-SC | $0.820 \pm 0.031$ |
| $M^3I$ | $\mathbf{0.832 \pm 0.029}$ |

Table 2: Performance comparison on Corel5K data set.

| Methods | Hamming loss ↓ | One-error ↓ | Coverage ↓ | Rank loss ↓ | Avg. prec. ↑ |
|---|---|---|---|---|---|
| MIMLBoost | 0.282 | 0.584 | 5.974 | 0.281 | 0.467 |
| MIMLSVM | 0.271 | 0.581 | 5.993 | 0.289 | 0.472 |
| DM | 0.243 | 0.575 | 5.512 | 0.236 | 0.541 |
| MildML | 0.238 | 0.569 | 5.107 | 0.233 | 0.554 |
| B2C | 0.275 | 0.580 | 5.823 | 0.283 | 0.470 |
| B2C-M | 0.270 | 0.562 | 5.675 | 0.241 | 0.493 |
| C2B | 0.224 | 0.545 | 5.032 | 0.229 | 0.565 |
| C2B-M | 0.216 | 0.538 | 4.912 | 0.218 | 0.572 |
| C2B-SC | 0.211 | 0.527 | 4.903 | 0.213 | 0.580 |
| $M^3I$ | **0.207** | **0.512** | **4.760** | **0.209** | **0.593** |

## 4 Experimental results

In this section, we experimentally evaluate the proposed $M^3I$ approach in image categorization tasks on three benchmark data sets: Object Recognition data set [2] which is a single-label image data set; and Corel5K data set [19] and PASCAL2010 data set [20] which are multi-label data sets.

### 4.1 Classification on single-label image data

Because the proposed $M^3I$ approach comprises two components, class specific metrics and significant coefficients, we implement four versions of our approach and evaluate their performances: (1) the simplest C2B distance, denoted as "C2B", computed by Eq. (3), in which no learning is involved; (2) C2B distance with class specific metrics, denoted as "C2B-M", computed by Eq. (4); (3) C2B distance with SCs, denoted as "C2B-SC" by Eq. (5) and set $M_k = I$; and (4) the C2B distance computed by proposed $M^3I$ approach using Eq. (5). We compare our methods against the following established MIL algorithms including (A) Diversity Density (DD) method [1], (B) DD-SVM method [2], (C) MIMLBoost method [3] and (D) MIMLSVM method [3]. We also compare our method to the two related methods, *i.e.*, (E) B2C method [9] and (F) B2C-M method [18]. These two methods are not MIL methods, therefore we consider each instance as an image descriptor following [9, 18]. We implement these methods following the original papers. The parameters of DD and DD-SVM are set according to the settings that resulted in the best performance [1,2]. The boosting rounds for MIMLBoost is set to 25 and for MIMLSVM we set $\gamma = 0.2$, which are same as in the experimental settings in [3]. For MIMLBoost and MIMLSVM, the top ranked class is regarded as the single-label prediction as in [3].

The classification accuracy is employed to measure the performance of the compared methods. Standard 5-fold cross-validation is performed and the classification accuracies averaged over all the 20 categories by the compared methods are presented in Table 1, where the means and standard deviations of the results in the 5 trials are reported and the best performances are bolded. The results in Table 1 show that the proposed $M^3I$ method clearly outperforms all other compared methods, which demonstrate the effectiveness of our method in single-label classification. Moreover, our $M^3I$ method is always better than its simplified versions, which confirms the usefulness of class specific metrics and SCs in MIL.

### 4.2 Classification on multi-label image data

Multi-label data refers to data sets in which an image can be associated with more than one semantic concept, which is more challenging but closer to real world applications than single-label data [21]. Thus, we evaluate the proposed method in multi-label image categorization tasks.

**Experimental settings.** We compare our approach to the following most recent MIML classification methods. (1) MIMLBoost method [3] and (2) MIMLSVM method [3] are designed for MIML classification, though they can also work with single-label multi-instance data as in last subsection. (3) Distance metric (DM) method [5] and (4) MildML method [6] learn a global distance metric from multi-instance data to compute B2B distances, therefore an additional classification step is

Table 3: Classification performance of comparison on PASCAL VOC 2010 data.

| Methods | Hamming loss ↓ | One-error ↓ | Coverage ↓ | Rank loss ↓ | Average precision ↑ |
|---|---|---|---|---|---|
| MIMLBoost | $0.183 \pm 0.020$ | $0.346 \pm 0.034$ | $1.034 \pm 0.075$ | $0.189 \pm 0.016$ | $0.472 \pm 0.023$ |
| MIMLSVM | $0.180 \pm 0.018$ | $0.349 \pm 0.029$ | $1.064 \pm 0.084$ | $0.181 \pm 0.014$ | $0.479 \pm 0.026$ |
| DM | $0.146 \pm 0.012$ | $0.307 \pm 0.024$ | $0.942 \pm 0.064$ | $0.167 \pm 0.013$ | $0.501 \pm 0.031$ |
| MildML | $0.139 \pm 0.011$ | $0.308 \pm 0.022$ | $0.951 \pm 0.058$ | $0.162 \pm 0.011$ | $0.504 \pm 0.029$ |
| B2C | $0.180 \pm 0.013$ | $0.343 \pm 0.020$ | $1.052 \pm 0.050$ | $0.148 \pm 0.023$ | $0.469 \pm 0.019$ |
| B2C-M | $0.177 \pm 0.010$ | $0.332 \pm 0.022$ | $0.993 \pm 0.049$ | $0.177 \pm 0.019$ | $0.502 \pm 0.023$ |
| C2B | $0.176 \pm 0.017$ | $0.326 \pm 0.027$ | $0.979 \pm 0.051$ | $0.168 \pm 0.020$ | $0.513 \pm 0.021$ |
| C2B-M | $0.145 \pm 0.014$ | $0.301 \pm 0.020$ | $0.966 \pm 0.046$ | $0.160 \pm 0.024$ | $0.509 \pm 0.026$ |
| C2B-SC | $0.137 \pm 0.010$ | $0.297 \pm 0.019$ | $0.925 \pm 0.035$ | $0.150 \pm 0.017$ | $0.527 \pm 0.016$ |
| $M^3I$ | $\mathbf{0.119 \pm 0.009}$ | $\mathbf{0.275 \pm 0.018}$ | $\mathbf{0.843 \pm 0.013}$ | $\mathbf{0.141 \pm 0.010}$ | $\mathbf{0.548 \pm 0.032}$ |

required. Following [5], we use citation-$K$NN [22] algorithm for classification, whose parameters are set as $R = 20$ and $C = 20$ as in [5]. We implement these method following their original works.

Corel5K data set has already been split into training set and test set, thus we train the compared methods using the 4500 training images and classify the 500 test images. We run 5-fold cross-validation on PASCAL VOC 2010 data set and report the "mean+std" performance over the 5 trails.

**Experimental results.** Because the two data sets used in our experiments are multi-label data sets, we measure the classification performances of the compared methods using five widely used multi-label evaluation metrics, as shown in Table 2 to 3, where "↓" indicates "the small the better" while "↑" indicates "the bigger the better". Details of these evaluation metrics can be found in [5, 23].

From Table 2 and 3, we can see that the proposed $M^3I$ method consistently outperforms the other methods, sometimes very significantly. Moreover, it is always better than its simplified versions.

Finally, we study the locally adaptive SCs learned for the training instances. In Figure 2, we show the SCs for several images in PASCAL VOC 2010 data set when they serve as training images. From the figures we can see that, a same object has different SCs when it is in different super-bags. For example, the instance of the person in the inner bounding box of the image in Figure 2(b) has comparably higher SC than the car instance in the outer bounding box when considering "person" class. In contrast, when it is in the super-bag of "car", its SC is lower than that of the car instance. These observations are consistent with our intuitions and theoretical analysis, because the person instance contribute considerably large in characterizing the "person" concept, whereas it contributes much less, or even possibly harmful, in characterizing the "car" concept. The same observations can also be seen on almost all the training images, which are not shown due to space limit. These interesting results provide concrete evidences to support the proposed $M^3I$ method's capability in revealing the semantic insight of a multi-instance image data set.

## 5 Conclusions

In this paper, we proposed a novel Maximum Margin Multi-Instance Learning ($M^3I$) method, which, instead of using the B2B distance as in many existing methods, approached MIL from a new perspective using the C2B distance to directly assess the relevance between classes and bags. Moreover, taking into account the two challenging properties of multi-instance data, high heterogeneity and weak label association, we further developed the C2B distance by introducing class specific distance metrics and locally adaptive SCs, which are learned by solving two convex maximum margin optimization problems. We applied the proposed $M^3I$ method in image categorization tasks on three benchmark data sets, one for single-label classification and two for multi-label classification. Encouraging experimental results by comparing our method to state-of-the-art MIL algorithms demonstrated its effectiveness.

**Acknowledgments**

This research was supported by NSF-IIS 1117965, NSFCCF-0830780, NSF-DMS-0915228, NSF-CCF-0917274.

# References

[1] O. Maron and A.L. Ratan. Multiple-instance learning for natural scene classification. In *ICML*, 1998.

[2] Y. Chen and J.Z. Wang. Image categorization by learning and reasoning with regions. *JMLR*, 5:913–939, 2004.

[3] Z.H. Zhou and M.L. Zhang. Multi-instance multi-label learning with application to scene classification. In *NIPS*, 2007.

[4] Z.J. Zha, X.S. Hua, T. Mei, J. Wang, G.J. Qi, and Z. Wang. Joint multi-label multi-instance learning for image classification. In *CVPR*, 2008.

[5] R. Jin, S. Wang, and Z.H. Zhou. Learning a distance metric from multi-instance multi-label data. In *CVPR*, 2009.

[6] M. Guillaumin, J. Verbeek, and C. Schmid. Multiple instance metric learning from automatically labeled bags of faces. In *ECCV*, 2010.

[7] H. Wang, F. Nie, and H. Huang. Learning instance specific distance for multi-instance classification. In *AAAI*, 2011.

[8] H. Wang, F. Nie, H. Huang, and Y. Yang. Learning frame relevance for video classification. In *ACM MM*, 2011.

[9] O. Boiman, E. Shechtman, and M. Irani. In defense of nearest-neighbor based image classification. In *CVPR*, 2008.

[10] A.W.M. Smeulders, M. Worring, S. Santini, A. Gupta, and R. Jain. Content-based image retrieval at the end of the early years. *IEEE TPAMI*, 22(12):1349–1380, 2002.

[11] H. Wang, H. Huang, and C. Ding. Image annotation using multi-label correlated Green's function. In *ICCV*, 2009.

[12] H. Wang, H. Huang, and C. Ding. Multi-label feature transform for image classifications. In *ECCV*, 2010.

[13] H. Wang, C. Ding, and H. Huang. Multi-label linear discriminant analysis. In *ECCV*, pages 126–139. Springer, 2010.

[14] H. Wang, H. Huang, and C. Ding. Image annotation using bi-relational graph of images and semantic labels. In *CVPR*, 2011.

[15] M. Schultz and T. Joachims. Learning a distance metric from relative comparisons. In *NIPS*, 2003.

[16] A. Frome, Y. Singer, and J. Malik. Image retrieval and classification using local distance functions. In *NIPS*, 2007.

[17] A. Frome, Y. Singer, F. Sha, and J. Malik. Learning globally-consistent local distance functions for shape-based image retrieval and classification. In *ICCV*, 2007.

[18] Z. Wang, Y. Hu, and L.T. Chia. Image-to-Class Distance Metric Learning for Image Classification. In *ECCV*, 2010.

[19] P. Duygulu, K. Barnard, J. De Freitas, and D. Forsyth. Object recognition as machine translation: Learning a lexicon for a fixed image vocabulary. In *ECCV*, 2002.

[20] M. Everingham, L. Van Gool, C. K. I. Williams, J. Winn, and A. Zisserman. The PASCAL Visual Object Classes Challenge 2010 (VOC2010) Results. http://pascallin.ecs.soton.ac.uk/challenges/VOC/voc2010/.

[21] H. Wang, C. Ding, and H. Huang. Multi-label classification: Inconsistency and class balanced k-nearest neighbor. In *AAAI*, 2010.

[22] J. Wang and J.D. Zucker. Solving the multiple-instance problem: A lazy learning approach. In *ICML*, 2000.

[23] R.E. Schapire and Y. Singer. BoosTexter: A boosting-based system for text categorization. *Machine learning*, 39(2):135–168, 2000.

